# Augmenting Feature-driven fMRI Analyses:
# Semi-supervised Learning and Resting State Activity

**Matthew B. Blaschko**
Visual Geometry Group
Department of Engineering Science
University of Oxford
blaschko@robots.ox.ac.uk

**Jacquelyn A. Shelton**
Max Planck Institute for Biological Cybernetics
Fakultät für Informations- und Kognitionswissenschaften
Universität Tübingen
jshelton@tuebingen.mpg.de

**Andreas Bartels**
Max Planck Institute for Biological Cybernetics
Centre for Integrative Neuroscience, Universität Tübingen
abartels@tuebingen.mpg.de

## Abstract

Resting state activity is brain activation that arises in the absence of any task, and is usually measured in awake subjects during prolonged fMRI scanning sessions where the only instruction given is to close the eyes and do nothing. It has been recognized in recent years that resting state activity is implicated in a wide variety of brain function. While certain networks of brain areas have different levels of activation at rest and during a task, there is nevertheless significant similarity between activations in the two cases. This suggests that recordings of resting state activity can be used as a source of unlabeled data to augment discriminative regression techniques in a semi-supervised setting. We evaluate this setting empirically yielding three main results: (i) regression tends to be improved by the use of Laplacian regularization even when no additional unlabeled data are available, (ii) resting state data seem to have a similar marginal distribution to that recorded during the execution of a visual processing task implying largely similar types of activation, and (iii) this source of information can be broadly exploited to improve the robustness of empirical inference in fMRI studies, an inherently data poor domain.

## 1 Introduction

In this work we study the use of resting state activity for the semi-supervised analysis of human fMRI studies. We wish to use resting state activity as an additional source of unlabeled data in a semi-supervised regression setting. We analyze the weights of a trained regressor to infer brain regions that are implicated in visual processing tasks. As the recording of human fMRI data is constrained by limits on the time a subject can safely remain in a scanner, and by the high demand for high-resolution scanning facilities, it is important to fully utilize available data. One source of such additional data is resting state activity, the brain activation that arises in the absence of any task. This data has been the subject of many studies in recent years, and has the important advantage of not being biased by a specific task. We show in this work that the marginal distribution of resting state activity is suitable to improve regression performance when employed for semi-supervised learning.

In neuroscience there has been a recent surge of interest in analyzing brain activity in more natural, complex settings, e.g. with volunteers viewing movies, in order to gain insight in brain processes and connectivity underlying more natural processing. The problem has been approached from dif-

ferent routes: linear regression was used to identify brain areas correlating with particular labels in the movie [2], the perceived content was inferred based on brain activity [23], data-driven methods were used to subdivide the brain into units with distinct response profiles [1], and correlation across subjects was used to infer stimulus-driven brain processes at different timescales [24]. Several pattern recognition techniques have previously been applied to fMRI data of brains, including support vector machines and Fisher linear discriminant analysis [26, 27, 29]. In [22], kernel canonical correlation analysis (KCCA) was applied to fMRI data from human subjects. We have recently applied a semi-supervised extension of KCCA to human fMRI data [32] where the unlabeled data source was given by the subjects viewing a movie for which the labels were not known. In this work, we explore the more realistic setting in which unlabeled data are available as a side product of other fMRI studies. This enables the more efficient use of available data, and obviates the necessity to waste scanner time and human labeling effort in order to produce sufficiently large data sets to achieve satisfactory results.

In Section 2 we discuss the generation and significance of resting state activity. We then discuss the statistical assumptions implicit in semi-supervised learning in Section 3. We present the experimental setup for data acquisition in Section 4, and discuss the semi-supervised regression model in Section 5. In Section 6, we show empirically that resting state activity is an effective source of unlabeled data for semi-supervised learning.

## 2 Resting State Activity

Resting state activity has attracted the attention of neuroscientists now for over a decade [8, 20]. It is defined as brain activation that arises in the absence of any task, and it is usually measured in awake subjects during prolonged fMRI scanning sessions, where no other instructions are given than to close the eyes and to do nothing. The basic idea is that spontaneous fluctuations of neural activity in the brain may reveal some fundamental characteristics of brain function. This may include functional aspects, but also structural ones.

For example, certain networks of areas have been shown to be more active at rest than during the execution of a task, leading to the hypothesis that these areas may be involved in maintaining the default state of the brain, performing mental house-keeping functions, such as monitoring own bodily states or the self [9, 30, 31], or being involved in intrinsic as opposed to extrinsic (i.e. stimulus-driven) tasks [17]. Additionally, spontaneous fluctuations of brain activity in particular brain regions have been shown to be directly correlated with metabolic activity and also with behavioural task performance, thus providing evidence that these fluctuations do not merely reflect artefacts of vascular perfusion, heart rate or breathing [9, 7]. Instead, evidence suggests that spontaneous activity changes reflect to some extent neural activity that may account for trial-to-trial variability of human behaviour [14, 28].

Resting state activity however also has structural implications, in that temporal correlations between spatially separate regions (functional connectivity) may be indicative of underlying neural communication between them, which in turn may be mediated by anatomical connections. Several studies have shown that homologue regions of the hemispheres (e.g. left and right motor cortex, Wernickes regions, etc) have high temporal correlations during rest [8, 3]. Also networks known to be anatomically connected, such as those belonging to the language network (Brocas area, Wernickes area, Geschwinds territory) within a given hemisphere show strong correlations during resting state, indicating that spontaneous activity (or activity driven by mental imagery, etc) in one region may affect others that are directly connected with it [21, 3]. Some recent studies also attempt to reveal that resting state connectivity directly relates to structural activity as revealed using diffusion tensor imaging [33, 19]. Finally, alterations in resting state activity patterns have recently been shown to be diagnostic for clinical conditions such as neuropsychiatric disorders [18], and have been shown to alter with increasing age [34].

However, the analysis of resting state activity poses a challenge, as it is not stimulus-driven, and is therefore difficult to analyze or to reveal using hypothesis-driven analysis methods. One common approach has been to reveal functional networks and their connectivity by measuring the temporal correlations of a seed region with the remaining voxels of the brain [8, 21, 3, 33, 19]. Another approach has been to apply data-driven spatio-temporal clustering methods such as independent component analysis (ICA) to reveal distinct functional areas or networks at rest [35, 1]. The over-

whelming evidence of these studies shows that groups of voxels, but also widespread networks of cortical areas that are co-engaged during task performances are also consistently co-activated during rest [35, 1, 12].

We provide an alternative, computationally driven approach to assess whether and to which extent externally driven functional networks coincide with spontaneous fluctuations during rest. We stimulated volunteers using natural movies, and measured resting state activity during the same session in separate runs that each lasted 20 min. Prior work has shown that natural movie viewing leads not only to wide-spread cortical activation, but also to a higher functional separation of distinct networks and areas compared to that obtained using traditional stimulation with controlled stimuli [23, 1, 17]. This is most likely so because distinct cortical regions responded each to distinct features occurring in the movie, thus revealing the functional division of labor in cortex [2, 4, 23].

In subsequent sections we show that semi-supervised learning algorithms improve when resting state data are added to aide feature-regression of movie-viewing data. This improvement indicates that a similar cortical structure underlies resting state data as that underlying movie-viewing data. These results thus fall in line with prior work demonstrating consistency of resting state networks across subjects [35, 12], and reveal that feature-driven activity during natural viewing induces a similar functional clustering as that occurring during rest. Importantly however, this approach may also be of other methodological interest, in that data obtained at rest may actually be used to augment the performance of feature-driven regression of stimulus-driven data.

## 3 Semi-supervised Learning

Semi-supervised learning makes use of a combination of labeled and unlabeled training points in order to better learn a mapping from an input space, $\mathcal{X}$ (in this case voxels recorded from fMRI), to an output space, $\mathcal{Y}$ (variables recording viewing conditions). Discriminative models typically attempt to infer a mapping $f : \mathcal{X} \rightarrow \mathcal{Y}$ based on properties the conditional distribution $p(y|x)$. In order to incorporate training data in $\mathcal{X}$ for which no correspondence is known to $\mathcal{Y}$, additional assumptions must be made about the properties of the joint distribution over $\mathcal{X} \times \mathcal{Y}$. This often gives semi-supervised learning more of a generative flavor in that we assume some properties of the joint distribution in order to better make use of the marginal distribution $p(x)$ [11].

There are several closely related assumptions employed in the development of semi-supervised learning algorithms, but we focus here on the manifold assumption [6]. We assume that our high dimensional data lie on a low dimensional manifold, and that changes in $p(y|x)$ vary slowly as measured by distances within the manifold. The additional unlabeled data in $\mathcal{X}$ allow us to better model the manifold on which the data lie.

In the case of fMRI acquired data, we expect that brain activity follow certain common patterns of activation. Furthermore, transitions between these patterns of activation will not be discontinuous. We can therefore be fairly confident in the assumption that the manifold assumption holds in principle. Of crucial importance, however, is that the distribution of the unlabeled samples not result in a degenerate marginal distribution with respect to the discriminative task at hand, that is to say that $p(y|x)$ be slowly varying as measured by distances measured within the manifold *estimated from labeled and unlabeled samples from $\mathcal{X}$*.

Theoretical accounts of semi-supervised learning frequently assume that *all* samples from $\mathcal{X}$ be drawn i.i.d. In practice, in a data poor domain, we may have to resort to a source of unlabeled data that is derived by a (slightly) different process than that of the labeled samples. As resting state data is a representative byproduct of the experimental design of fMRI studies, we explore the empirical performance of its employment as a source of unlabeled data. This gives us vital insight into whether the distribution of brain states is sufficiently similar to that of subjects who are performing a visual processing task, and suggests a general and powerful improvement to the design of fMRI studies by making use of this ready source of unlabeled information.

## 4 Data Acquisition

A Siemens 3TTIM scanner was used to acquire the fMRI data of 5 human volunteers and consisted of 350 time slices of 3-dimensional fMRI brain volumes. Time-slices were separated by 3.2 s (TR),

each with a spatial resolution of 46 slices (2.6 mm width, 0.4 mm gap) with 64x64 pixels of 3x3 mm, resulting in a spatial resolution of 3x3x3 mm. Each subject watched 2 movies of 18.5 min length, wherein one movie had labels indicating the continuous content of the movie (i.e. degree of visual contrast, or the degree to which a face was present, etc.) and the other remained unlabeled. The subjects additionally were recorded during a resting state of the same length of time. The imaging data were pre-processed using standard procedures using the Statistical Parametric Mapping (SPM) toolbox before analysis [15]. Included was a slice-time correction to compensate for acquisition delays between slices, a spatial realignment to correct for small head-movements, a spatial normalization to the SPM standard brain space (near MNI), and spatial smoothing using a Gaussian filter of 12 mm full width at half maximum (FWHM). Subsequently, images were skull-and-eye stripped and the mean of each time-slice was set to the same value (global scaling). A temporal high-pass filter with a cut-off of 512 s was applied, as well as a low-pass filter with the temporal properties of the hemodynamic response function (hrf), in order to reduce temporal acquisition noise.

For the movie with corresponding labels, the label time-series were obtained using two separate methods. First by using computer frame-by-frame analysis of the movie [4], and second using subjective ratings averaged across an independent set of five human observers [1]. The computer-derived labels indicated luminance change over time (temporal contrast), visual motion energy (i.e. the fraction of temporal contrast that can be explained by motion in the movie). The human-derived labels indicated the intensity of subjectively experienced color, and the degree to which faces and human bodies were present in the movie. In prior studies, each of these labels had been shown to correlate with brain activity in particular and distinct sets of areas specialized to process the particular label in question [1, 4].

## 5 Regression Model

We have applied a semi-supervised Laplacian regularized ridge regression framework to learn our discriminant function. We assume multivariate data $x_i \in \mathcal{X}$ with associated labels $y_i \in \mathbb{R}$, for $i = 1, \ldots, n$, although the setting is directly extensible to arbitrary input and output domains [10]. Ridge regression is classically formulated as

$$\text{argmin}_w \sum_i \left( y_i - \langle x_i, w \rangle \right)^2 + \lambda \|w\|^2, \tag{1}$$

where $x$ and $y$ are assumed to be zero mean [25]. This is equivalent to maximizing (Tikhonov regularized) correlation between $y$ and the projection of $x$ onto $w$ [16]. In order to extend this to the semi-supervised setting [11], we assume the manifold assumption and employ Laplacian regularization [37, 38, 5, 36, 6]. We assume that we have $p_x$ additional unlabeled training samples and use the variable $m_x = n + p_x$ for notational convenience. We denote the design matrix of labeled data as $X$ and that of labeled and unlabeled data $\hat{X}$. We can now write our Laplacian regularized objective function as

$$\text{argmin}_w (y - Xw)^T (y - Xw) + \lambda \|w\|^2 + \frac{\gamma}{m_x^2} w^T \hat{X}^T \mathcal{L} \hat{X} w \tag{2}$$

where $\mathcal{L}$ is an empirical graph Laplacian [6].

The two regularization parameters, $\lambda$ and $\gamma$, are set using a model selection step. We have employed a variant of the model selection used in [32], which employs a grid search to maximize the difference in objective functions between a randomized permutation of the correspondences between $x$ and $y$ and the unpermuted data. We have used a symmetric normalized graph Laplacian where the weights are given by a Gaussian function with the bandwidth set to the median distance between training data points

$$\mathcal{L} = I - D^{-\frac{1}{2}} S D^{-\frac{1}{2}}, \tag{3}$$

where $S$ is a similarity matrix and D is a diagonal matrix whose entries are the row sums of $S$.

We have primarily chosen this regression model for its simplicity. Provided the manifold assumption holds for our source of data, and that the conditional distribution, $p(y|x)$, is slowly varying as measured by the manifold estimated from both labeled and unlabeled data, we can expect that semi-supervised Laplacian regularization will improve results across a range of loss functions and output spaces.

Table 1: Mean holdout correlations for *motion* in the five subjects across all experiments. For a description of the experiments, see Section 5. In all cases, semi-supervision from resting state activity (Exp C) improves over regression using only fully labeled data (Exp A).

|       | Sub 1          | Sub 2          | Sub 3          | Sub 4           | Sub 5          |
|-------|----------------|----------------|----------------|-----------------|----------------|
| Exp A | $-0.008 \pm 0.12$ | $-0.08 \pm 0.07$ | $-0.08 \pm 0.04$ | $-0.06 \pm 0.07$ | $-0.08 \pm 0.08$ |
| Exp B | $-0.02 \pm 0.17$ | $-0.03 \pm 0.10$ | $0.01 \pm 0.09$ | $-0.02 \pm 0.04$ | $-0.03 \pm 0.08$ |
| Exp C | $0.12 \pm 0.06$ | $0.10 \pm 0.10$ | $0.17 \pm 0.14$ | $0.012 \pm 0.09$ | $0.06 \pm 0.12$ |
| Exp D | $0.09 \pm 0.09$ | $0.10 \pm 0.14$ | $0.15 \pm 0.15$ | $0.04 \pm 0.04$ | $0.02 \pm 0.11$ |
| Exp E | $0.11 \pm 0.10$ | $0.11 \pm 0.15$ | $0.12 \pm 0.09$ | $0.11 \pm 0.08$ | $0.16 \pm 0.15$ |

Table 2: Mean holdout correlations for *human body* in the five subjects across all experiments. For a description of the experiments, see Section 5. In all cases, semi-supervision from resting state activity (Exp C) improves over regression using only fully labeled data (Exp A).

|       | Sub 1          | Sub 2           | Sub 3          | Sub 4          | Sub 5          |
|-------|----------------|-----------------|----------------|----------------|----------------|
| Exp A | $0.13 \pm 0.17$ | $-0.003 \pm 0.12$ | $0.09 \pm 0.11$ | $0.06 \pm 0.14$ | $0.12 \pm 0.17$ |
| Exp B | $0.16 \pm 0.16$ | $0.16 \pm 0.22$  | $0.28 \pm 0.15$ | $0.16 \pm 0.20$ | $0.21 \pm 0.16$ |
| Exp C | $0.36 \pm 0.17$ | $0.29 \pm 0.16$  | $0.42 \pm 0.15$ | $0.30 \pm 0.12$ | $0.40 \pm 0.06$ |
| Exp D | $0.34 \pm 0.09$ | $0.30 \pm 0.14$  | $0.38 \pm 0.25$ | $0.25 \pm 0.11$ | $0.35 \pm 0.11$ |
| Exp E | $0.35 \pm 0.22$ | $0.37 \pm 0.17$  | $0.45 \pm 0.08$ | $0.33 \pm 0.14$ | $0.43 \pm 0.05$ |

As our data consist of (i) recordings from a completely labeled movie, (ii) recordings from resting state activity, and (iii) recordings from an unlabeled movie, we are able to employ several variants of semi-supervision in the above framework:

- A: In this variant, we employ only fully supervised data and use the regression given by Equation (1).
- B: We also use only fully supervised data in this variant, but we employ Laplacian regularization in addition to Tikhonov regularization (Equation (2)).
- C: We introduce semi-supervision from resting state activity.
- D: In this variant, semi-supervision comes from the unlabeled movie. This allows us to evaluate the effects of semi-supervision from data that are designed to be drawn from the same distribution as our labeled data.
- E: Finally, we combine the unlabeled data from both resting state activity and from the unlabeled movie.

# 6 Experimental Results

In order to evaluate the performance of the regression model with different semi-supervised variants, we have performed five fold cross validation. For each fold, we measure the correlation between the projected data and its associated labels. We have performed these experiments across five different subjects with three different output variables. Table 1 shows the test correlations for all subjects and experiments for the *motion* output variable, while Table 2 shows results for the *human body* variable, and Table 3 for the *language* variable. Wilcoxon signed-rank tests have shown significant

Table 3: Mean holdout correlations for *language* in the five subjects across all experiments. For a description of the experiments, see Section 5. In all cases, semi-supervision from resting state activity (Exp C) improves over regression using only fully labeled data (Exp A).

|       | Sub 1          | Sub 2           | Sub 3          | Sub 4           | Sub 5           |
|-------|----------------|-----------------|----------------|-----------------|-----------------|
| Exp A | $0.10 \pm 0.13$ | $0.10 \pm 0.10$  | $0.11 \pm 0.14$ | $-0.03 \pm 0.17$ | $-0.03 \pm 0.11$ |
| Exp B | $0.15 \pm 0.17$ | $-0.05 \pm 0.09$ | $0.06 \pm 0.23$ | $0.14 \pm 0.18$  | $0.03 \pm 0.14$  |
| Exp C | $0.35 \pm 0.10$ | $0.15 \pm 0.11$  | $0.42 \pm 0.03$ | $0.07 \pm 0.17$  | $0.10 \pm 0.13$  |
| Exp D | $0.27 \pm 0.17$ | $0.29 \pm 0.14$  | $0.34 \pm 0.20$ | $0.08 \pm 0.11$  | $-0.03 \pm 0.11$ |
| Exp E | $0.34 \pm 0.17$ | $0.22 \pm 0.15$  | $0.30 \pm 0.18$ | $0.24 \pm 0.15$  | $0.07 \pm 0.19$  |

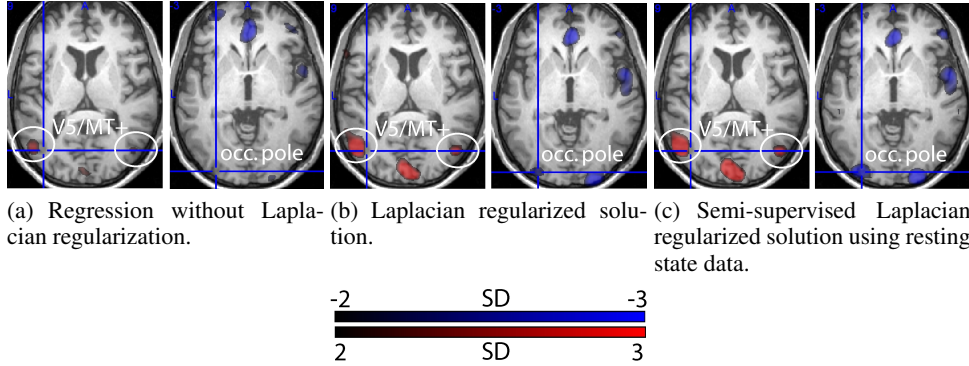

(a) Regression without Laplacian regularization.

(b) Laplacian regularized solution.

(c) Semi-supervised Laplacian regularized solution using resting state data.

Figure 1: Illustration of weight maps obtained for the visual motion feature in experiments A, B, and D. Transverse slices are shown through a single subjects T1-weighted structural image with superimposed weight-maps, colored in red for positive weights (left column), and colored in blue for negative weights (right column). The positive weight maps (left column) reveal the motion processing area V5/MT+, as well as posterior in the midline a part of peripheral early visual area V1 (not labelled). The negative weight maps reveal a reduction of BOLD signal in the occipital poles (the foveal representation of early visual areas V1-V3). Both results are in agreement with the findings reported in a prior study[4].

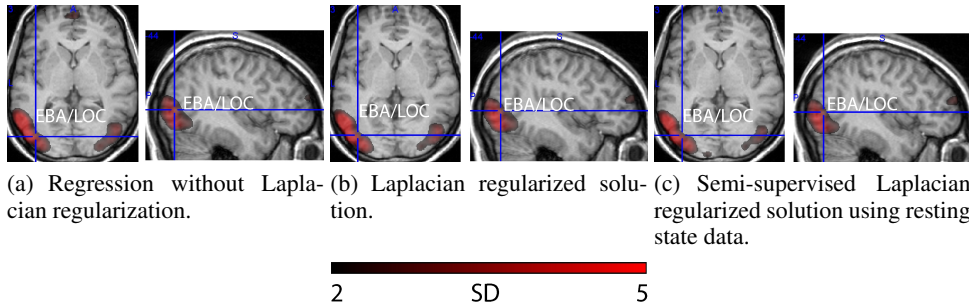

(a) Regression without Laplacian regularization.

(b) Laplacian regularized solution.

(c) Semi-supervised Laplacian regularized solution using resting state data.

Figure 2: Illustration of weight maps for the human body feature. Weight maps (in red) are show on transverse (left) and sagittal (right) brain sections of a single subject. Activity involves the object-responsive lateral occipital cortex (LOC) extending dorsally into region responsive to human bodies, dubbed extrastriate body area (EBA) [13]. The weights in all experiments are very strong for this feature (see colorbar), and nearly no difference in the extent of activation is visible across experiments.

improvement between ridge regression and semi-supervised Laplacian regularization with confidence $> 95\%$ for all variables. We also provide a qualitative evaluation of the results in the form of a map of the significant weights onto slices shown through single subjects. Figure 1 shows the weights for the *motion* variable, Figure 2 for the *human body* variable, and Figure 3 for the *language* variable.

## 7  Discussion

One can observe several trends in Tables 1-3. First, we notice that the results for experiment A are not satisfactory. Correlations appear to be non-existent or low, and show high variation across subjects. We conclude that the labeled training data alone are not sufficient to learn a reliable regressor for these learning problems. The results in experiment B are mixed. For some subjects and variables performance improved, but it is not consistent. We expect that this indicates non-linearity in the data, but that the labeled data alone are not sufficient to accurately estimate the manifold. We see consistent improvement in experiment C over experiment A. This supports the primary hypothesis of this work – that the marginal distribution of resting state activity in combination of that from the visual

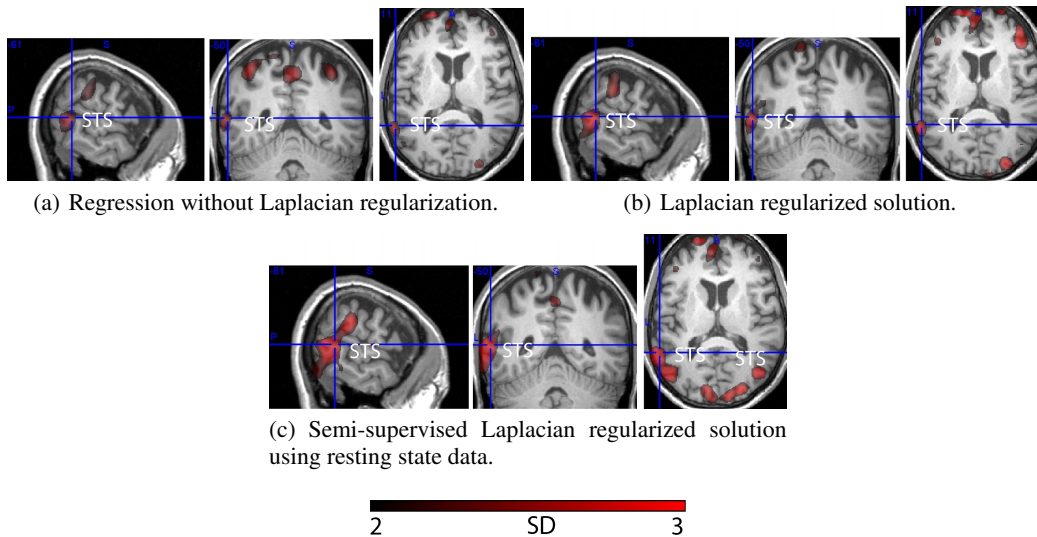

(a) Regression without Laplacian regularization.    (b) Laplacian regularized solution.

(c) Semi-supervised Laplacian regularized solution using resting state data.

Figure 3: Illustration of weight maps obtained for the language feature across the different experiments. Weight maps (in red) are superimposed on sagittal, coronal and transverse sections of a single subjects brain. The activation associated to this feature involved the superior temporal sulcus (STS), extending anteriorly to include parts of Wernickes speech processing area, and posterior and ventrally (increasing with experiments A, B and D) object-responsive region LOC, involved in analyzing facial features (in accord with the findings in [2]).

processing task allows us to robustly estimate a manifold structure that improves regression performance. The results for experiment C and D are similar, with neither data source dominating the other. As the unlabeled data for experiment D were generated specifically to match the distribution of the labeled data, we conclude that resting state activity gives a similar increase in performance to semi-supervised learning with i.i.d. data. Finally, the setup in experiment E – in which we use both sources of semi-supervised data – performs similarly on average to that in experiments C and D. We conclude that the two sources of unlabeled data may not hold complimentary data, indicating that a wholesale replacement of one source by another is an effective strategy.

The feature-weight maps shown in Figures 1-3 were all in accord with established findings in neuroscience, in that distinct features such as visual motion, the perception of human bodies or of language correlated with activation of distinct brain regions, such as V5+/MT+, the lateral occipital complex (LOC) and the extrastriate body area (EBA), as well as regions of the STS and Wernickes area, respectively. These findings have now been established in studies using controlled stimuli, as well as those showing movie-clips to volunteers [13, 2, 4, 23].

Here we asked whether using semi-supervised learning methods can improve a feature-driven analysis when adding data obtained in the resting state. The motivation for this stems from prior studies that suggest a functionally relevant involvement of cortical regions during rest. Data-driven analyses of resting state activity reveals a similar functional architecture that can also be observed during stimulus-driven activity, and which can be reproducibly found across subjects [12, 35]. In addition, also the functional connectivity between distinct regions appears to be physiologically plausible at rest [21, 8, 20], and in fact is very similar to the functional connectivity observed during viewing of movies [3]. Taken together, these findings would suggest that resting state activity may in theory be able to augment in a non-biased way datasets obtained in a functional setting. At the same time, if resting state data were indeed found to augment results of feature-driven analyses, this would form an important finding, as it would directly indicate that resting state activity indeed is similar in its nature to that induced by stimulus-driven settings. Our findings indeed appear to show such an effect, as is illustrated in Figures 1-3. For example, the activation of visual motion responsive cortex V5+/MT+ clearly increased in experiments A-C. Note that this was not only reflected in the positive weights, but also in the negative ones; in complete consistency with the findings reported in [4] even the negative involvement of foveal visual representations with increase of visual motion

became amplified with the addition of resting state data. Similar findings concerned the cortical regions involved in the perception of language. However, this augmenting effect was not observed in all subjects for all features Figure 2 for example shows a subject in whom the human body feature obtained very high weights already in the most basic analysis, and no augmentation was apparent in the weight maps for the more complex analyses, perhaps reflecting a saturation effect. Since the resting state is not well-defined, it may also be that particular internal states, sleepiness, etc. would not guarantee augmenting in all datasets.

All in all however our results show that adding resting state data can indeed augment findings obtained in stimulus-inducing settings. This method may therefore be useful for the increasing number of imaging centres acquiring resting state data for completely different purposes, which may then be used to augment functional data, entirely free of cost in terms of scan time. An even more promising prospect however is that also the baseline or rest condition within stimulus-driven sessions may be used to augment the results obtained in the stimulus conditions. This may be especially valuable, since almost all imaging sessions contain baseline conditions, that are often not used for further analysis, but take up considerable amount of scan time.

Apart from the above, application-orientated considerations, our findings also provide new evidence that brain-states during rest which are difficult to characterize indeed resemble those during exposure to complex, natural stimulation. Our approach is therefore an extension of prior attempts to characterize the complex, rich, yet difficult to characterize brain activation during the absence of externally driven stimulation.

## 8 Conclusions

In this work, we have proposed the use of resting state data as a source for the unlabeled component of semi-supervised learning for fMRI studies. Experimental results show that one of the primary assumptions of semi-supervised learning, the manifold assumption, holds well for this data, and that the marginal distribution of unlabeled resting state data is observed to augment that of labeled data to consistently improve regression performance. Semi-supervised Laplacian regularization is a widely applicable regularization technique that can be added to many kinds of machine learning algorithms. As we have shown that the basic assumptions of semi-supervised learning hold for this kind of data, we expect that this approach would work on these other discriminant/regression methods as well, including kernel logistic regression, support vector machines, and kernel canonical correlation analysis.

As data acquisition and the manual labeling of stimulus data are expensive components of brain imaging, the benefits of exploiting additional unlabeled data are clear. Resting state data is a promising source as there are no task specific biases introduced. In future work we intend to further study the properties of the distribution of resting state activity. We also intend to pursue cross subject studies. If brain activity is consistent across subjects for the specific task measured by a study, a large cross subject sample of resting state data may be employed to improve results.

**Acknowledgments**

The first author is supported by the Royal Academy of Engineering through a Newton International Fellowship. The second author is supported by an ACM-W scholarship.

## References

[1] A. Bartels and S. Zeki. The chronoarchitecture of the human brain–natural viewing conditions reveal a time-based anatomy of the brain. *NeuroImage*, 22(1):419 – 433, 2004.

[2] A. Bartels and S. Zeki. Functional brain mapping during free viewing of natural scenes. *Human Brain Mapping*, 21(2):75–85, 02/01/ 2004.

[3] A. Bartels and S. Zeki. Brain dynamics during natural viewing conditions–a new guide for mapping connectivity in vivo. *NeuroImage*, 24(2):339 – 349, 2005.

[4] A. Bartels, S. Zeki, and N. K. Logothetis. Natural vision reveals regional specialization to local motion and to contrast-invariant, global flow in the human brain. *Cereb. Cortex*, pages bhm107+, July 2007.

[5] M. Belkin and P. Niyogi. Semi-supervised learning on riemannian manifolds. *Machine Learning*, 56(1-3):209–239, 2004.

[6] M. Belkin, P. Niyogi, and V. Sindhwani. Manifold Regularization: A Geometric Framework for Learning from Labeled and Unlabeled Examples. *JMLR*, 7:2399–2434, 2006.

[7] M. Bianciardi, M. Fukunaga, P. van Gelderen, S. G. Horovitz, J. A. de Zwart, and J. H. Duyn. Modulation of spontaneous fMRI activity in human visual cortex by behavioral state. *NeuroImage*, 45(1):160–168, 2009.

[8] B. Biswal, Z. F. Yetkin, V. M. Haughton, and J. S. Hyde. Functional connectivity in the motor cortex of resting human brain using echo-planar mri. *Magnetic Resonance in Medicine*, 34(4):537–541, 1995.

[9] B. B. Biswal, J. Van Kylen, and J. S. Hyde. Simultaneous assessment of flow and bold signals in resting-state functional connectivity maps. *NMR Biomed*, 10(4-5):165–170, 1997.

[10] M. B. Blaschko, C. H. Lampert, and A. Gretton. Semi-supervised laplacian regularization of kernel canonical correlation analysis. In *ECML PKDD '08: Proceedings of the 2008 European Conference on Machine Learning and Knowledge Discovery in Databases I*, pages 133–145. Springer-Verlag, 2008.

[11] O. Chapelle, B. Schölkopf, and A. Zien, editors. *Semi-Supervised Learning*. MIT Press, Cambridge, MA, 2006.

[12] J. S. S. Damoiseaux, S. A. R. B. A. Rombouts, F. Barkhof, P. Scheltens, C. J. J. Stam, S. M. M. Smith, and C. F. F. Beckmann. Consistent resting-state networks across healthy subjects. *Proc Natl Acad Sci U S A*, August 2006.

[13] P. E. Downing, Y. Jiang, M. Shuman, and N. Kanwisher. A Cortical Area Selective for Visual Processing of the Human Body. *Science*, 293(5539):2470–2473, 2001.

[14] M. D. Fox, A. Z. Snyder, J. M. Zacks, and M. E. Raichle. Coherent spontaneous activity accounts for trial-to-trial variability in human evoked brain responses. *Nature Neuroscience*, 9(1):23–25, December 2005.

[15] K. Friston, J. Ashburner, S. Kiebel, T. Nichols, and W. Penny, editors. *Statistical Parametric Mapping: The Analysis of Functional Brain Images*. Academic Press, 2007.

[16] T. V. Gestel, J. A. K. Suykens, J. D. Brabanter, B. D. Moor, and J. Vandewalle. Kernel canonical correlation analysis and least squares support vector machines. In *ICANN '01: Proceedings of the International Conference on Artificial Neural Networks*, pages 384–389, London, UK, 2001. Springer-Verlag.

[17] Y. Golland, S. Bentin, H. Gelbard, Y. Benjamini, R. Heller, Y. Nir, U. Hasson, and R. Malach. Extrinsic and Intrinsic Systems in the Posterior Cortex of the Human Brain Revealed during Natural Sensory Stimulation. *Cereb. Cortex*, 17(4):766–777, 2007.

[18] M. Greicius. Resting-state functional connectivity in neuropsychiatric disorders. *Current opinion in neurology*, 24(4):424–430, August 2008.

[19] M. D. Greicius, K. Supekar, V. Menon, and R. F. Dougherty. Resting-State Functional Connectivity Reflects Structural Connectivity in the Default Mode Network. *Cereb. Cortex*, 19(1):72–78, 2009.

[20] R. Gur, L. Mozley, P. Mozley, S. Resnick, J. Karp, A. Alavi, S. Arnold, and R. Gur. Sex differences in regional cerebral glucose metabolism during a resting state. *Science*, 267(5197):528–531, 1995.

[21] M. Hampson, B. S. Peterson, P. Skudlarski, J. C. Gatenby, and J. C. Gore. Detection of functional connectivity using temporal correlations in mr images. *Hum Brain Mapp*, 15(4):247–262, April 2002.

[22] D. R. Hardoon, J. Mourão-Miranda, M. Brammer, and J. Shawe-Taylor. Unsupervised Analysis of fMRI Data Using Kernel Canonical Correlation. *NeuroImage*, 37(4):1250–1259, 2007.

[23] U. Hasson, Y. Nir, I. Levy, G. Fuhrmann, and R. Malach. Intersubject Synchronization of Cortical Activity During Natural Vision. *Science*, 303(5664):1634–1640, 2004.

[24] U. Hasson, E. Yang, I. Vallines, D. J. Heeger, and N. Rubin. A hierarchy of temporal receptive windows in human cortex. *J. Neurosci.*, 28(10):2539–2550, March 2008.

[25] T. Hastie, R. Tibshirani, and J. Friedman. *The Elements of Statistical Learning: Data Mining, Inference, and Prediction*. Springer, 2001.

[26] J. V. Haxby, M. I. Gobbini, M. L. Furey, A. Ishai, J. L. Schouten, and P. Pietrini. Distributed and Overlapping Representations of Faces and Objects in Ventral Temporal Cortex. *Science*, 293(5539):2425–2430, 2001.

[27] J.-D. Haynes and G. Rees. Decoding mental states from brain activity in humans. *Nature Reviews Neuroscience*, 7(7):523–534, July 2006.

[28] C. Kelly, L. Uddin, B. Biswal, F. Castellanos, and M. Milham. Competition between functional brain networks mediates behavioral variability. *NeuroImage*, 39(1):527–537, January 2008.

[29] S.-P. Ku, A. Gretton, J. Macke, and N. K. Logothetis. Comparison of pattern recognition methods in classifying high-resolution bold signals obtained at high magnetic field in monkeys. *Magnetic Resonance Imaging*, 26(7):1007 – 1014, 2008.

[30] M. E. Raichle, A. M. MacLeod, A. Z. Snyder, W. J. Powers, D. A. Gusnard, and G. L. Shulman. A default mode of brain function. *Proc Natl Acad Sci U S A*, 98(2):676–682, January 2001.

[31] M. E. E. Raichle and A. Z. Z. Snyder. A default mode of brain function: A brief history of an evolving idea. *Neuroimage*, 37(4):1083–1090, October 2007.

[32] J. Shelton, M. Blaschko, and A. Bartels. Semi-supervised subspace analysis of human functional magnetic resonance imaging data. Technical Report 185, Max Planck Institute for Biological Cybernetics, 2009.

[33] P. Skudlarski, K. Jagannathan, V. D. Calhoun, M. Hampson, B. A. Skudlarska, and G. Pearlson. Measuring brain connectivity: Diffusion tensor imaging validates resting state temporal correlations. *NeuroImage*, 43(3):554 – 561, 2008.

[34] M. C. Stevens, G. D. Pearlson, and V. D. Calhoun. Changes in the interaction of resting-state neural networks from adolescence to adulthood. *Human Brain Mapping*, 8(30):2356–2366, 2009.

[35] V. G. van de Ven, E. Formisano, D. Prvulovic, C. H. Roeder, and D. E. Linden. Functional connectivity as revealed by spatial independent component analysis of fmri measurements during rest. *Hum Brain Mapp*, 22(3):165–178, July 2004.

[36] D. Zhou, O. Bousquet, T. N. Lal, J. Weston, and B. Schölkopf. Learning with local and global consistency. In *Advances in Neural Information Processing Systems 16*, 2004.

[37] X. Zhu and Z. Ghahramani. Learning from labeled and unlabeled data with label propagation. Technical Report CMU-CALD-02-107, Carnegie Mellon University, 2002.

[38] X. Zhu, Z. Ghahramani, and J. Lafferty. Semi-supervised learning using gaussian fields and harmonic functions. In *International Conference on Machine Learning*, pages 912–919, 2003.

